# Efficient Unsupervised Learning for Localization and Detection in Object Categories

**Nicolas Loeff, Himanshu Arora**
ECE Department
University of Illinois at
Urbana-Champaign
{loeff,harora1}@uiuc.edu

**Alexander Sorokin, David Forsyth**
Computer Science Department
University of Illinois at
Urbana-Champaign
{sorokin2,daf}@uiuc.edu

## Abstract

We describe a novel method for learning templates for recognition and localization of objects drawn from categories. A generative model represents the configuration of multiple object parts with respect to an object coordinate system; these parts in turn generate image features. The complexity of the model in the number of features is low, meaning our model is much more efficient to train than comparative methods. Moreover, a variational approximation is introduced that allows learning to be orders of magnitude faster than previous approaches while incorporating many more features. This results in both accuracy and localization improvements. Our model has been carefully tested on standard datasets; we compare with a number of recent template models. In particular, we demonstrate state-of-the-art results for detection and localization.

## 1 Introduction

Building appropriate object models is central to object recognition, which is a fundamental problem in computer vision. Desirable characteristics of a model include good representation of objects, fast and efficient learning algorithms that require as little supervised information as possible. We believe an appropriate representation of an object should allow for both detection of its presence and localization ('where is it?'). So far the quality of object recognition in the literature has been measured by its detection performance only. Viola and Jones [1] present a fast object detection system boosting Haar filter responses. Another effective discriminative approach is that of a *bag of keypoints* [2, 3]. It is based on clustering image patches using appearance only, disregarding geometric information. The performance for detection in this algorithm is among the state of the art. However as no geometry cues are used during training, features that do not belong to the object can be incorporated into the object model. This is similar to classic *overfitting* and typically leads to problems in object localization.

Weber *et. al.* [4] represent an object as a constellation of parts. Fergus *et. al.* [5] extend the model to account for variability in appearance. The model encodes a template as a set of feature-generating parts. Each part generates at most one feature. As a result the complexity is determined by hardness of part-feature assignment. Heuristic search is used to approximate the solution, but feasible problems are limited to 7 parts with 30 features.

Agarwal and Roth [6] learn using SNoW a classifier on a sparse representation of patches extracted around interesting points in the image. In [7], Leibe and Schiele use a voting scheme to predict object configuration from locations of individual patches. Both approaches provide localization, but require manually localizing the objects in training images. Hillel et. al. [8] independently proposed an approach similar to ours. Their model however has higher learning complexity and inferior detection performance despite being of discriminative nature.

In this paper, we present a generative probabilistic model for detection and localization of objects that can be efficiently learnt with minimal supervision. The first crucial property of the model is that it represents the configuration of multiple object parts with respect to an unobserved, *abstract* object root (unlike [9, 10], where an "object root" is chosen as one of the visible parts of the object). This simplifies localization and allows our model to overcome occlusion and errors in feature extraction. The model also becomes symmetric with respect to visible parts. The second crucial assumption of the model is that a single part can generate multiple features in the image (or none). This may seem counterintuitive, but keypoint detectors generally detects several features around *interesting* areas. This hypothesis also makes an explicit model for part occlusion unnecessary: instead occlusion of a part means implicitly that no feature in the image is produced by it.

These assumptions allow us to model all features in the image as being emitted independently conditioned on the object center. As a result the complexity of inference in our model is **linear** in the number of parts of the model and the number of features in the image, obviating the exponential complexity of combinatoric assignments in other approaches [4, 5, 11]. This means our model is much easier than constellation models to train using Expectation Maximization (EM), which enables the use of more features and more complex models with resulting improvements in both accuracy and localization. Furthermore we introduce a variational (mean-field) approximation during learning that allows it to be hundreds of times faster than previous approaches, with no substantial loss of accuracy.

## 2  Model

Our model of an object category is a template that generates features in the image. Each image is represented as a set $\{f_j\}$ of $F$ features extracted with the scale-saliency point detector [13]. Each feature is described by its location and appearance. Feature extraction and representation will be detailed in section 3. As described in the introduction, we hypothesize that given the object center all features are generated *independently*: $p^{obj}(f_1, .., f_F) = \sum_{o_c} P(o_c) \prod_j p(f_j|o_c)$. The abstract object center - which does not generate any features - is represented by a *hidden* random variable $o_c$. For simplicity it takes values in a discrete grid of size $N_x \times N_y$ inside the image and $o_c$ is assumed to be a priori uniformly distributed in its domain.

Conditioned on the object center, each feature is generated by a mixture of $P$ parts plus a background part. A set of *hidden* variables $\{\omega_{ij}\}$ represents which part $(i)$ produced feature $f_j$. These variables $\omega_{ij}$ then take values $\{0, 1\}$ restricted to $\sum_{i=1}^{P+1} \omega_{ij} = 1$. In other words, $\omega_{ij} = 1$ means feature $j$ was produced by part $i$; each part can produce multiple features, each feature is produced by only one part. The distribution of a feature conditioned on the object center is then $p(f_j|o_c) = \sum_i p(f_j, w_{ij} = 1|o_c) = \sum_i p(f_j|w_{ij} = 1, o_c)\pi_i$, where $\pi_i$ is the prior emission probability of part $i$. $\pi_i$ is subject to $\sum_{i=1}^{P+1} \pi_i = 1$.

Each part has a location distribution with respect to the object center corresponding to a two dimensional full covariance Gaussian, $p_L^i(x|o_c)$. The appearance (see section 3 for details) of a part does not depend on the configuration of the object; we consider two models :

**Gaussian Model (G)** Appearance $p_A^i$ is modeled as a $k$ dimensional diagonal covariance Gaussian distribution.

**Local Topic Model (LT)** Appearance $p_A^i$ is modeled as a multinomial distribution on a previously learnt $k$-word image patch dictionary. This can be considered as a local topic model.

Let $\theta$ denote the set of parameters. The complete data likelihood (joint distribution) for image $n$ in the object model is then,

$$P_\theta^{obj}\left(\{\omega_{ij}\}, o_c, \{f_j\}\right) = \prod_{o_c'} \left\{ \prod_{j,i} \left\{ p_L^i(f_j|o_c') p_A^i(f_j)\pi_i \right\}^{[\omega_{ij}=1]} P(o_c') \right\}^{[o_c=o_c']} \tag{1}$$

where $[expr]$ is one if $expr$ is true and zero otherwise. Marginalizing, the probability of the observed image in the object model is then,

$$P_\theta^{obj}\left(\{f_j\}\right) = \sum_{o_c} P(o_c) \prod_{j'} \left\{ \sum_i P(f_{j'}, \omega_{ij'} = 1 | o_c) \right\} \tag{2}$$

The background model assumes all features are produced independently, with uniform location on the image. In the G model of appearance, the appearance is modeled with a $k$ dimensional full covariance matrix Gaussian distribution. In the LT model, we use a multinomial distribution on the $k$-word image patch dictionary to model the appearance.

## 2.1 Learning

The maximum-likelihood solution for the parameters of the above model does not have a closed form. In order to train the model the parameters are computed numerically using the approach of [14], minimizing a free-energy $F_e$ associated with the model that is an upper bound on the negative log-likelihood. Following [14], we denote $v = \{f_j\}$ as the set of visible and $h = \{o_c, \omega_{ij}\}$ as the set of hidden variables. Let $D_{KL}$ be the K-L divergence:

$$F_e(Q, \theta) = D_{KL}\{Q(h)||P_\theta(h|v)\} - \log P_\theta(v) = \int_h Q(h) \log \frac{Q(h)}{P_\theta(h,v)} dh \tag{3}$$

In this bound, $Q(h)$ can be a *simpler* approximation of the posterior probability $P_\theta(h|v)$, that is used to compute estimates and update parameters. Minimizing eq. 3 with respect to $Q$ and $\theta$ under different restrictions, produces a range of algorithms including exact EM, variational learning and others [14]. Table 2.1 shows sample updates and complexity of these algorithms and comparison to other relevant work.

The background model is learnt before the object model is trained. As assumed earlier, for Gaussian appearance model the background appearance model is a single gaussian, whose mean and variance are estimated as the sample mean and covariance. For the Local Topic model, the multinomial distribution is estimated as the sample histogram. The model for background feature location is uniform and does not have any parameters.

**EM Learning for the Object model:** In the E-step, the set of parameters $\theta$ is fixed and $F_e$ is minimized with respect to $Q(h)$ without restrictions. This is equivalent to computing the actual posteriors in EM [14, 15]. In this case the optimal solution factorizes as $Q(h) = Q(o_c)Q(\omega_{ij}|o_c) = P(o_c|v)P(\omega_{ij}|o_c,v)$. In the M-step, $F_e$ is minimized with respect to the parameters $\theta$ using the current estimate of $Q$. Due to the conditional independence introduced in the model, inference is tractable and thus the E-step can be computed efficiently. The overall complexity of inference is $O(FP \cdot N_x N_y)$.

| Model | Update for $\mu_L^i$ | Complexity | Time (F,P) |
|---|---|---|---|
| Fergus *et al.* | N/A | $F^P$ | 36 hrs  (30, 7) |
| Model (EM) | $\mu_L^i \leftarrow \dfrac{\sum_n \sum_{o_c} Q(o_c) \sum_j Q(\omega_{ji}|o_c)\{x_L^j - o_c\}}{\sum_n \sum_{o_c} Q(o_c) \sum_j Q(\omega_{ji}|o_c)}$ | $FP \cdot N_x N_y$ | 3 hrs  (50, 30) |
| (Variational) | $\mu_L^i \leftarrow \dfrac{\sum_n \left\{\sum_j Q(\omega_{ji})x_L^j - \sum_{o_c} Q(o_c)o_c\right\}}{\sum_n \sum_{o_c} Q(o_c) \sum_j Q(\omega_{ji})}$ | $FP + N_x N_y$ | 3 mins  (100, 30) |

Table 1: An example of an update, overall complexity and convergence time for our models and [5], for different number of features per image ($F$) and number of parts in the object model ($P$). There is an increase in speed of several orders of magnitude with respect to [5] on similar hardware.

**Variational Learning:** In this approach a mean field approximation of $Q$ is considered; in the E-step the parameters $\theta$ are fixed and $F$ is minimized with respect to $Q$ under the restriction that it factorizes as $Q(h) = Q(o_c)Q(w_{ij})$. This corresponds to a decoupling of location ($o_c$) and part-feature assignment ($w_{ij}$) in the approximation ($Q$) of the posterior $P_\theta(h|v)$. In the M-step $\theta$ is fixed and the free energy $F_e$ is minimized with respect to this (mean field) version of $Q$. A comparison between EM and Variational updates of the mean in location $\mu_L^i$ of a part is shown in table 2.1. The overall complexity of inference is now $O(FP) + O(N_x N_y)$; this represents orders of magnitude of speedup with respect to the already efficient EM learning. The impact on performance of the variational approximation is discussed in section 4.

## 2.2 Detection and localization

For detection of object presence, a natural decision rule is the likelihood ratio test. After the models are learnt, for each test image $P_\theta^{obj}(\{f_j\})/P^{bg}(\{f_j\})$ is compared to a threshold to make the decision. Once the presence of the object is established, the most likely location is given by the MAP estimate of $o_c$. We assign parts in the model to the object if they exhibit consistent appearance and location. To remove model parts representing background we use a threshold on the entropy of the appearance distribution for the LT model (the determinant of the covariance in location for the G model). The MAP estimate of which features in the image are assigned (marginalizing over the object center) to parts in the model determines the support of the object. Bounding boxes include all keypoints assigned to the object and means of all model parts belonging to the object even if no keypoint is observed to be produced by such part. This explicitly handles occlusion (fig. 1).

## 3 Experimental setup

The performance of the method depends on the feature detector making consistent extraction in different instances of objects of the same type. We use the scale-saliency interest point detector proposed in [13]. This method selects regions exhibiting unpredictable characteristics over both location and scale. The $F$ regions with highest saliency over the image provide the features for learning and recognition. After the keypoints are detected, patches are extracted around this points and scale-normalized. A SIFT descriptor [16] (without orientation) is obtained from these patches. For model G, due to the high dimensionality of resulting space, PCA is performed choosing $k = 15$ components to represent the appearance of a feature. For model LT, we instead cluster the appearance of features in the original SIFT space with a gaussian mixture model with $k = 250$ components and use the most likely cluster as feature appearance representation.

For all experiments we use $P = 30$ parts. The number of features is $F = 50$ for G model and $F = 100$ for LT model, $N_x \times N_y = 238$. We test our approach on the Caltech 5 dataset: faces, motorbikes, airplanes, spotted cats vs. Caltech background and cars rear 2001 vs. cars background [5]. We initialize appearance and location of the parts with $P$ randomly chosen features from the training set. The stopping criterion is the change in $F_e$.

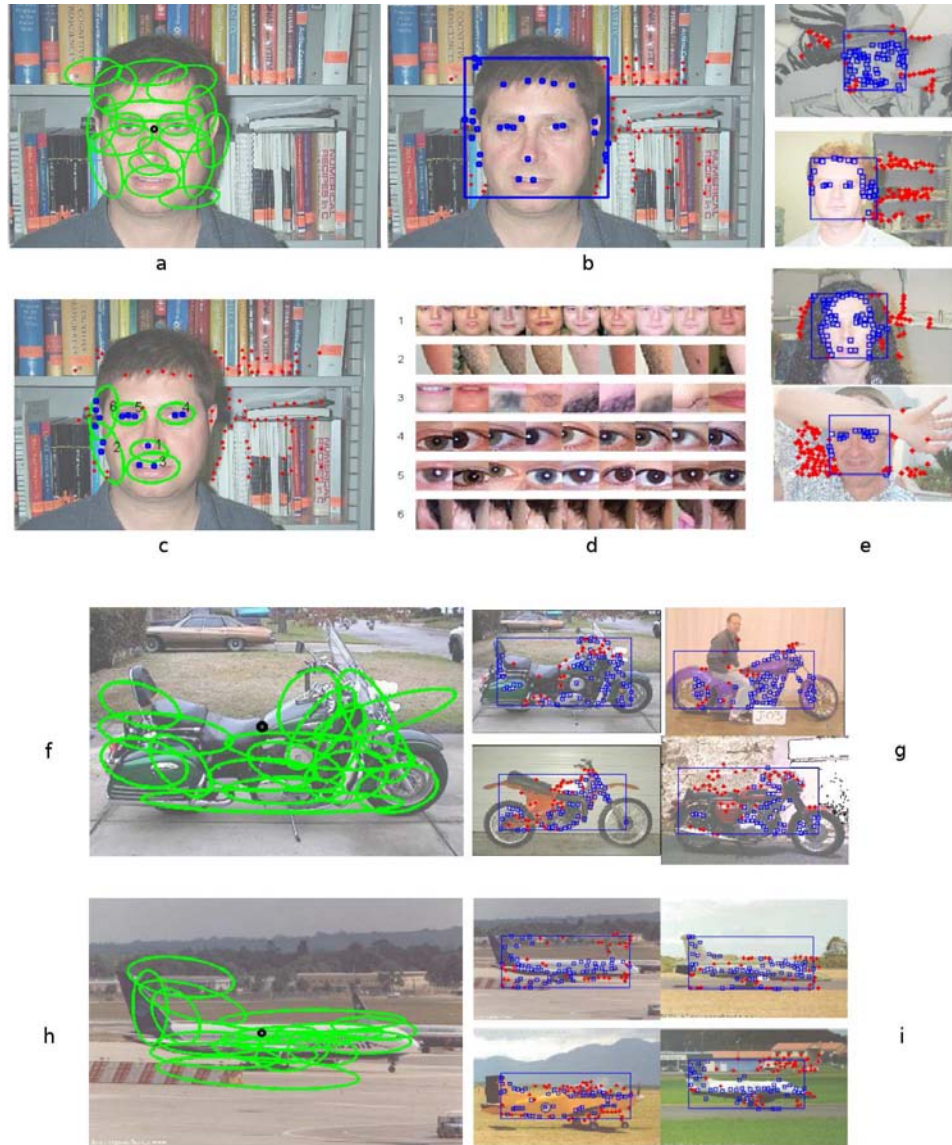

Figure 1: Local Topic model for faces, motorbikes and airplanes datasets [5]. In **(a)** the most likely location of the object center is plotted as a black circle. With respect to this reference, the spatial distribution (2D gaussian) of each part associated with the object is plotted in green. In **(b)** the centers of all features extracted are depicted. Blue ones are assigned by the model to the object, and red ones to the background. The bounding box is plotted in blue. Image **(c)** shows how many features in the image are assigned to the same part (a property of our model, not shared by [5]): six parts are chosen, their spatial distribution is plotted (green), and the features assigned to them are depicted in blue. Eyes (4,5), mouth (3) and left ear (6) have multiple assignments each. For each these parts, image **(d)** image shows the best matches in features extracted from the dataset. Note that the local topic model can learn parts uniform in appearance (i.e. eyes) but also more complex parts (i.e. the mouth part includes moustaches, beards and chins). The G appearance model and [5] do not have this property. The images **(e)** show the robustness of the method in cases with occlusion, missed detections and one caricature of a face. Images **(f)** and **(g)** show plots for motorbikes, and **(h)** and **(i)** for airplanes.

## 4 Results

**Detection:** Although we believe that localization is an essential performance criterion, it is useless if the approach cannot detect objects. Figure 2 depicts equal error rate detection performance for our models and [5, 3, 8]. We can not compare our range of performance (for train/test splits), shown on the plot, because this data is not available for other approaches. Our method is robust to initialization (the variance for starting points is negligible compared to train/test split variance). The results show higher detection performance of all our algorithms compared to the generative model presented in [5]. The local topic (LT) model performs better than the model presented in [8]. The purely discriminative approach presented in [3] shows higher detection performance with different ("optimal combination") features, but performs worse for the features we are using. The LT model showed consistently higher detection performance than the Gaussian (G) model. For both LT and G models the variational approximations showed similar discriminative power to that of the respective exact models. Unlike [5, 3], our model currently is not scale invariant. Nevertheless the probabilistic nature of the model allows for some tolerance to scale changes.

In datasets of manageable size, it is inevitable that the background is correlated with the object. The result is that most modern methods that infer the template form partially supervised data can tend to model some background parts as lying on the object (see figure 4). Doing so tends to *increase* detection performance. It is reasonable to expect this increase will not persist in the face of a dramatic change in background. One symptom of this phenomenon (as in classical overfitting) is that methods that detect very well may be bad at localization, because they cannot separate the object from background. We are able to avoid this difficulty by predicting object extent conditioned on detection using only a subset of parts known to have relatively low variance in location or appearance, given the object center. We do not yet have an estimate of the increase in detection rate resulting from *overfitting*. This is a topic of ongoing research. In our opinion, if a method can detect but performs poorly at localization, the reason may be overfitting.

**Localization:** Previous work on localization required aligned images (bounding boxes) or segmentation masks [7, 6]. A novel property of our model is that it learns to localize the object and determine its spatial extent without supervision. Figure 1 shows learned models and examples of localization. There is no standard measure to evaluate localization performance in an unsupervised setting. In such a case, the object center can be learnt at any position in the image, provided that this position is consistent across all images. We thus use as our performance measure, the standard deviation of estimated object centers and bounding boxes (obtained as in §2.2), after normalizing the estimates of each image to a coordinate system in which the ground truth bounding box is a unit square $(0,0) - (1,1)$.

As a baseline we use the rectified center of the image. All objects of interest in both airplane and motorbike datasets are centered in the image. As a result the baseline is a good predictor of the object center and is hard to beat. However in the faces dataset there is much more variation in location; then the advantage of our approach becomes clear. Figure 3 shows the scatterplot of normalized object centers and bounding boxes. The table in figure 2 shows the localization performance results using the proposed metric.

**Variational approximation comparison:** Unusually for a variational approximation it is possible to compare it to the exact model; the results are excellent especially for the G model. This is consistent with our observation that during learning the variational approximation is good in this case (the free energy bound appears tight). On the other hand, for the LT model, the variational bound is loose during learning and localization performance is equivalent, but slightly lower than that of exact LT model. This may be explained by the fact that gaussian appearance model is less flexible then the topic model and thus G model can better tolerate decoupling of location and appearance.

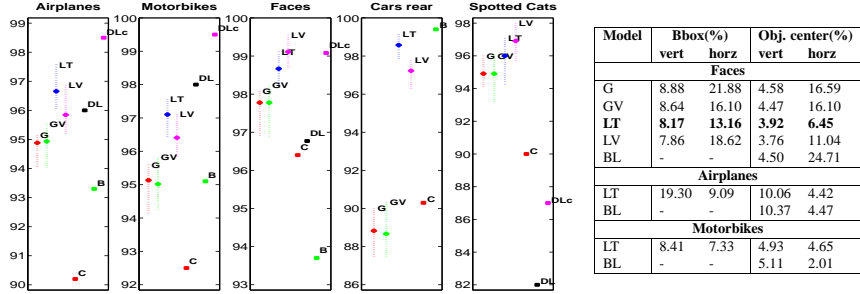

| Model | Bbox(%) | | Obj. center(%) | |
|---|---|---|---|---|
| | vert | horz | vert | horz |
| Faces | | | | |
| G | 8.88 | 21.88 | 4.58 | 16.59 |
| GV | 8.64 | 16.10 | 4.47 | 16.10 |
| **LT** | **8.17** | **13.16** | **3.92** | **6.45** |
| LV | 7.86 | 18.62 | 3.76 | 11.04 |
| BL | - | - | 4.50 | 24.71 |
| Airplanes | | | | |
| LT | 19.30 | 9.09 | 10.06 | 4.42 |
| BL | - | - | 10.37 | 4.47 |
| Motorbikes | | | | |
| LT | 8.41 | 7.33 | 4.93 | 4.65 |
| BL | - | - | 5.11 | 2.01 |

Figure 2: Plots on the left show detection performance on Caltech 5 datasets [5]. Equal error rate is reported. The original performance of constellation model [5] is denoted by C. We denote by DLc the performance (best in literature) reported by [3] using an optimal combination of feature types, and by DL the performance using our features. The performance of [8] is denoted by B. We show performance for our G model (G), LT model (L) and their variational approximations (GV) and (LV) respectively. We report median performance ($\times$) over 20 runs and performance range excluding 10% best and 10% worst runs. On the right we show localization performance for all models on Faces dataset and performance of the best model (LT) on all datasets. Standard deviation is reported in percentage units with respect to the ground truth bounding box. For bounding boxes we average the standard deviation in each direction. BL denotes baseline performance.

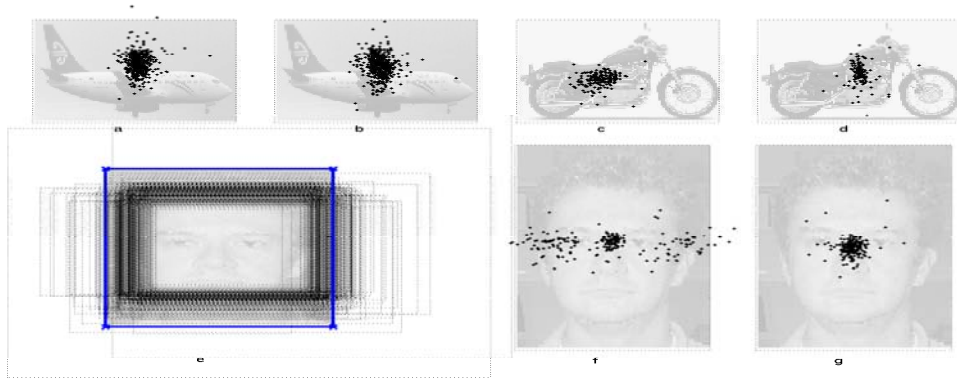

Figure 3: The airplane and motorbike datasets are aligned. Thus the image center baseline (**b**), (**d**) performs well there. Our localization performs similarly (**a**), (**c**). There is more variation in location in faces dataset. Scatterplot (**f**) shows the baseline performance and (**g**) shows the performance of our model. (**e**) shows the bounding boxes computed by our approach (LT model). Object centers and bounding boxes are rectified using the ground truth bounding boxes (blue). No information about location or spatial extent of the object is given to the algorithm.

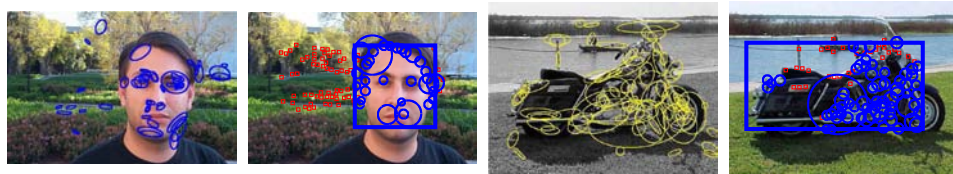

Figure 4: Approaches like [3] do not use geometric constraints during learning. Therefore, correlation between background and object in the dataset is incorporated into the object model. In this case the ellipses represent the features that are used by the algorithm in [3] to decide the presence of a face and motorbike (**left** images taken from [3]). On the other hand, our model (**right** images) can estimate the location and support of the object, even though no information about it is provided during learning. Blue circles represent the features assigned by the model to the face, the red points are centers of features assigned to background (plot for Local Topic Model).

# 5   Conclusions and future work

We have presented a novel model for object categories. Our model allows efficient unsupervised learning, bringing the learning time to a few hours for full models and to minutes for variational approximations. The significant reduction in complexity allows to handle many more parts and features than comparable algorithms. The detection performance of our approach compares favorably to the state of the art even when compared to purely discriminative approaches. Also our model is capable of learning the spatial extent of the objects without supervision, with good results.

This combination of fast learning and ability to localize is required to tackle challenging problems in computer vision. Among the most interesting applications we see unsupervised segmentation, learning, detection and localization of multiple object categories, deformable objects and objects with varying aspects.

## References

[1] P. Viola and M. Jones. Rapid object detection using a boosted cascade of simple features. *Proc. of CVPR*, pages 511–518, 2001.

[2] G. Csurka, C. Dance, L. Fan, and C. Bray. Visual Categorization with Bags of Keypoints. In *Workshop on Stat. Learning in Comp. Vision, ECCV*, pages 1–22, 2004.

[3] G. Dorkó and C. Schmid. Object class recognition using discriminative local features. Submitted to *IEEE trans. on PAMI*, 2004.

[4] M. Weber, M. Welling, and P. Perona. Unsupervised Learning of Models for Recognition. *Proc. of ECCV (1)*, pages 18–32, 2000.

[5] R. Fergus, P. Perona, and A. Zisserman. Object Class Recognition by Unsupervised Scale-Invariant Learning. *Proc. of CVPR*, pages 264–271, 2003.

[6] S. Agarwal and D. Roth. Learning a sparse representation for object detection. In *Proc. of ECCV*, volume 4, pages 113–130, Copenhagen, Denmark, May 2002.

[7] B. Leibe, A. Leonardis, and B. Schiele. Combined object categorization and segmentation with an implicit shape model. In *Workshop on Stat. Learning in Comp. Vision*, pages 17–32, May 2004.

[8] A. B. Hillel, T. Hertz, and D. Weinshall. Efficient learning of relational object class models. In *Proc. of ICCV*, pages 1762–1769, October 2005.

[9] R. Fergus, P. Perona, and A. Zisserman. A sparse object category model for efficient learning and exhaustive recognition. In *Proc. of CVPR*, pages 380–387, june 2005.

[10] D. Crandall, P. Felzenszwalb, and D. Huttenlocher. Spatial Priors for Part-Based Recognition using Statistical Models. In *Proc. of CVPR*, pages 10–17, 2005.

[11] L. Fei-Fei, R. Fergus, and P. Perona. Learning generative visual models from few training examples an incremental bayesian approach tested on 101 object categories. In *Workshop on Generative-Model Based Vision*, Washington, DC, June 2004.

[12] A. Opelt, M. Fussenegger, A. Pinz, and P. Auer. Generic object recognition with boosting. Technical Report TR-EMT-2004-01, EMT, TU Graz, Austria, 2004. Submitted to the *IEEE Trans. on PAMI*.

[13] T. Kadir and M. Brady. Saliency, Scale and Image Description. *IJCV*, 45(2):83–105, 2001.

[14] B. Frey and N. Jojic. A Comparison of Algorithms for Inference and Learning in Probabilistic Graphical Models. *IEEE Trans. on PAMI*, 27(9):1392–1416, 2005.

[15] R. Neal and G. Hinton. A view of the EM algorithm that justifies incremental, sparse, and other variants. In M. I. Jordan, editor, *Learning in graphical models*, pages 355–368. MIT Press, Cambridge, MA, USA, 1999.

[16] D. Lowe. Distinctive image features from scale-invariant keypoints. *IJCV*, 60(2):91–110, 2004.
